# Proximal Deep Structured Models

**Shenlong Wang**
University of Toronto
slwang@cs.toronto.edu

**Sanja Fidler**
University of Toronto
fidler@cs.toronto.edu

**Raquel Urtasun**
University of Toronto
urtasun@cs.toronto.edu

## Abstract

Many problems in real-world applications involve predicting continuous-valued random variables that are statistically related. In this paper, we propose a powerful deep structured model that is able to learn complex non-linear functions which encode the dependencies between continuous output variables. We show that inference in our model using proximal methods can be efficiently solved as a feed-foward pass of a special type of deep recurrent neural network. We demonstrate the effectiveness of our approach in the tasks of image denoising, depth refinement and optical flow estimation.

## 1 Introduction

Many problems in real-world applications involve predicting a collection of random variables that are statistically related. Over the past two decades, graphical models have been widely exploited to encode these interactions in domains such as computer vision, natural language processing and computational biology. However, these models are shallow and only a log linear combination of hand-crafted features is learned [34]. This limits the ability to learn complex patterns, which is particularly important nowadays as large amounts of data are available, facilitating learning.

In contrast, deep learning approaches learn complex data abstractions by compositing simple non-linear transformations. In recent years, they have produced state-of-the-art results in many applications such as speech recognition [17], object recognition [21], stereo estimation [38], and machine translation [33]. In some tasks, they have been shown to outperform humans, e.g., fine grained categorization [7] and object classification [15].

Deep neural networks are typically trained using simple loss functions. Cross entropy or hinge loss are used when dealing with discrete outputs, and squared loss when the outputs are continuous. Multi-task approaches are popular, where the hope is that dependencies of the output will be captured by sharing intermediate layers among tasks [9].

*Deep structured models* attempt to learn complex features by taking into account the dependencies between the output variables. A variety of methods have been developed in the context of predicting discrete outputs [7, 3, 31, 39]. Several techniques unroll inference and show how the forward and backward passes of these deep structured models can be expressed as a set of standard layers [1, 14, 31, 39]. This allows for fast end-to-end training on GPUs.

However, little to no attention has been given to deep structured models with continuous valued output variables. One of the main reasons is that inference (even in the shallow model) is much less well studied, and very few solutions exist. An exception are Markov random fields (MRFs) with Gaussian potentials, where exact inference is possible (via message-passing) if the precision matrix is positive semi-definite and satisfies the spectral radius condition [36]. A family of popular approaches convert the continuous inference problem into a discrete task using particle methods [18, 32]. Specific solvers have also been designed for certain types of potentials, *e.g.* polynomials [35] and piecewise convex functions [37].

Proximal methods are a popular solution to perform inference in continuous MRFs when the potentials are non-smooth and non-differentiable functions of the outputs [26]. In this paper, we show that proximal methods are a special type of recurrent neural networks. This allows us to efficiently train a wide family of deep structured models with continuous output variables end-to-end on the GPU. We show that learning can simply be done via back-propagation for any differentiable loss function. We demonstrate the effectiveness of our algorithm in the tasks of image denoising, depth refinement and optical flow and show superior results over competing algorithms on these tasks.

## 2 Proximal Deep Structured Networks

In this section, we first introduce continuous-valued deep structured models and briefly review proximal methods. We then propose proximal deep structured models and discuss how to do efficient inference and learning in these models. Finally we discuss the relationship with previous work.

### 2.1 Continuous-valued Deep Structured Models

Given an input $\mathbf{x} \in \mathcal{X}$, let $\mathbf{y} = (y_1, ..., y_N)$ be the set of random variables that we are interested in predicting. The output space is a product space of all the elements: $\mathbf{y} \in \mathcal{Y} = \prod_{i=1}^{N} \mathcal{Y}_i$, and the domain of each individual variable $y_i$ is a closed subset in real-valued space, *i.e.* $\mathcal{Y}_i \subset \mathbb{R}$. Let $E(\mathbf{x}, \mathbf{y}; \mathbf{w}) : \mathcal{X} \times \mathcal{Y} \times \mathbb{R}^K \to \mathbb{R}$ be an energy function which encodes the problem that we are interested in solving. Without loss of generality we assume that the energy decomposes into a sum of functions, each depending on a subset of variables

$$E(\mathbf{x}, \mathbf{y}; \mathbf{w}) = \sum_i f_i(y_i, \mathbf{x}; \mathbf{w}_u) + \sum_\alpha f_\alpha(\mathbf{y}_\alpha, \mathbf{x}; \mathbf{w}_\alpha) \qquad (1)$$

where $f_i(y_i; \mathbf{x}, \mathbf{w}) : \mathcal{Y}_i \times \mathcal{X} \to \mathbb{R}$ is a function that depends on a single variable (i.e., unary term) and $f_\alpha(\mathbf{y}_\alpha) : \mathcal{Y}_\alpha \times \mathcal{X} \to \mathbb{R}$ depends on a subset of variables $\mathbf{y}_\alpha = (y_i)_{i \in \alpha}$ defined on a domain $\mathcal{Y}_\alpha \subset \mathcal{Y}$. Note that, unlike standard MRF models, the functions $f_i$ and $f_\alpha$ are non-linear functions of the parameters.

The energy function is parameterized in terms of a set of weights $\mathbf{w}$, and learning aims at finding the value of these weights which minimizes a loss function. Given an input $\mathbf{x}$, inference aims at finding the best configuration by minimizing the energy function:

$$\mathbf{y}^* = \arg\min_{\mathbf{y} \in \mathcal{Y}} \sum_i f_i(y_i, \mathbf{x}; \mathbf{w}_u) + \sum_\alpha f_\alpha(\mathbf{y}_\alpha, \mathbf{x}; \mathbf{w}_\alpha) \qquad (2)$$

Finding the best scoring configuration $\mathbf{y}^*$ is equivalent to maximizing the posteriori distribution: $p(\mathbf{y}|\mathbf{x}; \mathbf{w}) = \frac{1}{Z(\mathbf{x};\mathbf{w})} \exp(-E(\mathbf{x}, \mathbf{y}|\mathbf{w}))$, with $Z(\mathbf{x}; \mathbf{w})$ the partition function.

Standard multi-variate deep networks (*e.g.*, FlowNet [11]) have potential functions which depend on a single output variable. In this simple case, inference corresponds to a forward pass that predicts the value of each variable independently. This can be interpreted as inference in a graphical model with only unary potentials $f_i$.

In the general case, performing inference in MRFs with continuous variables involves solving a very challenging numerical optimization problem. Depending on the structure and properties of the potential functions, various methods have been proposed. For instance, particle methods perform approximate inference by performing message passing on a series of discrete MRFs [18, 32]. Exact inference is possible for a certain type of MRFs, *i.e.*, Gaussian MRFs with positive semi-definite precision matrix. Efficient dedicated algorithms exist for a restricted family of functions, *e.g.*, polynomials [35]. If certain conditions are satisfied, inference is often tackled by a group of algorithms called proximal methods [26]. In this section, we will focus on this family of inference algorithms and show that they are a particular type of recurrent net. We will use this fact to efficiently train deep structured models with continuous outputs.

### 2.2 A Review on Proximal Methods

Next, we briefly discuss proximal methods, and refer the reader to [26] for a thorough review. Proximal algorithms are very generally applicable, but they are particularly successful at solving non-smooth, non-differentiable, or constrained problems. Their base operation is evaluating the proximal operator of a function, which involves solving a small convex optimization problem that

often admits a closed-form solution. In particular, the proximal operator $\text{prox}_f(x_0) : \mathbb{R} \to \mathbb{R}$ of a function is defined as

$$\text{prox}_f(x_0) = \arg\min_y (y - x_0)^2 + f(y)$$

If $f$ is convex, the fixed points of the proximal operator of $f$ are precisely the minimizers of $f$. In other words, $\text{prox}_f(x^*) = x^*$ iff $x^*$ minimizes $f$. This fix-point property motivates the simplest proximal method called the proximal point algorithm which iterates $x^{(n+1)} = \text{prox}_f(x^{(n)})$. All the proximal algorithms used here are based on this fix-point property. Note that even if the function $f(\cdot)$ is not differentiable (*e.g.*, $\ell_1$ norm) there might exist a closed-form or easy to compute proximal operator.

While the original proximal operator was designed for the purpose of obtaining the global optimum in convex optimization, recent work has shown that proximal methods work well for non-convex optimization as long as the proximal operator exists [20, 30, 5].

For multi-variate optimization problems the proximal operator might not be trivial to obtain (*e.g.*, when having high-order potentials). In this case, a widely used solution is to decompose the high-order terms into small problems that can be solved through proximal operators. Examples of this family of algorithms are half-quadratic splitting [13], alternating direction method of multipliers [12] and primal-dual methods [2] . In this work, we focus on the non-convex multi-variate case.

## 2.3 Proximal Deep Structured Models

In order to apply proximal algorithms to tackle the inference problem defined in Eq. (2), we require the energy functions $f_i$ and $f_\alpha$ to satisfy the following conditions:

1. There exist functions $h_i$ and $g_i$ such that $f_i(y_i, \mathbf{x}; \mathbf{w}) = g_i(y_i, h_i(\mathbf{x}, \mathbf{w}))$, where $g_i$ is a distance function [1];

2. There exists a closed-form proximal operator for $g_i(y_i, h_i(\mathbf{x}; \mathbf{w}))$ wrt $y_i$.

3. There exist functions $h_\alpha$ and $g_\alpha$ such that $f_\alpha(y_\alpha, \mathbf{x}; \mathbf{w})$ can be re-written as $f_\alpha(y_\alpha, \mathbf{x}; \mathbf{w}) = h_\alpha(\mathbf{x}; \mathbf{w})g_\alpha(\mathbf{w}_\alpha^T \mathbf{y}_\alpha)$.

4. There exists a proximal operator for either the dual or primal form of $g_\alpha(\cdot)$.

A fairly general family of deep structured models satisfies these conditions. Our experimental evaluation will demonstrate the applicability in a wide variety of tasks including depth refinement, image denoising as well as optical flow. If our potential functions satisfy the conditions above, we can rewrite our objective function as follows:

$$E(\mathbf{x}, \mathbf{y}; \mathbf{w}) = \sum_i g_i(y_i, h_i(\mathbf{x}; \mathbf{w})) + \sum_\alpha h_\alpha(\mathbf{x}; \mathbf{w})g_\alpha(\mathbf{w}_\alpha^T \mathbf{y}_\alpha) \tag{3}$$

In this paper, we make the important observation that each iteration of most existing proximal solvers contain five sub-steps: (i) compute the locally linear part; (ii) compute the proximal operator $\text{prox}_{g_i}$; (iii) deconvolve; (iv) compute the proximal operator $\text{prox}_{g_\alpha}$; (v) update the result through a gradient descent step. Due to space restrictions, we show primal-dual solvers in this section, and refer the reader to the supplementary material for ADMM, half-quadratic splitting and the proximal gradient method.

The general idea of primal dual solvers is to introduce auxiliary variables $\mathbf{z}$ to decompose the high-order terms. We can then minimize $\mathbf{z}$ and $\mathbf{y}$ alternately through computing their proximal operator. In particular, we can transform the primal problem in Eq. (3) into the following saddle point problem

$$\min_{\mathbf{y} \in \mathcal{Y}} \max_{\mathbf{z} \in \mathcal{Z}} \sum_i g_i(y_i, h_i(\mathbf{x}, \mathbf{w}_u)) - \sum_\alpha h_\alpha(\mathbf{x}, \mathbf{w})g_\alpha^*(\mathbf{z}_\alpha) + \sum_\alpha h_\alpha(\mathbf{x}, \mathbf{w})\langle \mathbf{w}_\alpha^T \mathbf{y}_\alpha, \mathbf{z}_\alpha \rangle \tag{4}$$

where $g_\alpha^*(\cdot)$ is the convex conjugate of $g_\alpha(\cdot)$: $g_\alpha^*(\mathbf{z}^*) = \sup\{\langle \mathbf{z}^*, \mathbf{z} \rangle - g_\alpha(\mathbf{z}) | \mathbf{z} \in \mathcal{Z}\}$ and the convex conjugate of $g_\alpha^*$ is $g_\alpha$ itself, if $g_\alpha(\cdot)$ is convex.

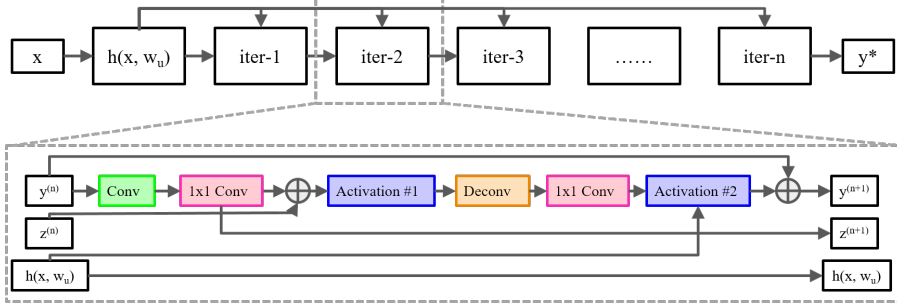

Figure 1: The whole architecture (top) and one iteration block (bottom) of our proximal deep structured model.

The primal-dual method solves the problem in Eq. (4) by iterating the following steps: (i) fix $\mathbf{y}$ and minimize the energy wrt $\mathbf{z}$; (ii) fix $\mathbf{z}$ and minimize the energy wrt $\mathbf{y}$; (iii) conduct a Nesterov extrapolation gradient step. These iterative computation steps are:

$$
\begin{cases}
z_\alpha^{(t+1)} &= \operatorname{prox}_{g_\alpha^*}\left(z_\alpha^{(t)} + \dfrac{\sigma_\rho}{h_\alpha(\mathbf{x};\mathbf{w})}\mathbf{w}_\alpha^T \bar{\mathbf{y}}_\alpha^{(t)}\right) \\
y_i^{(t+1)} &= \operatorname{prox}_{g_i, h_i(\mathbf{x},\mathbf{w})}\left(y_i^{(t)} - \dfrac{\sigma_\tau}{h_\alpha(\mathbf{x};\mathbf{w})}\mathbf{w}_{\cdot,i}^{*T}\mathbf{z}^{(t+1)}\right) \\
\bar{y}_i^{(t+1)} &= y_i^{(t+1)} + \sigma_{ex}\left(y_i^{(t+1)} - y_i^{(t)}\right)
\end{cases}
\tag{5}
$$

where $\mathbf{y}^{(t)}$ is the solution at the $t$-th iteration, $\mathbf{z}^{(t)}$ is an auxiliary variable and $h(\mathbf{x}, \mathbf{w}_u)$ is the deep unary network. Note that different functions $g_i$ and $g_\alpha$ in (3) have different proximal operators.

It is not difficult to see that the inference process in Eq. (5) can be written as a feed-forward pass in a recurrent neural network by stacking multiple computation blocks. In particular, the first step is a convolution layer and the third step can be considered as a deconvolution layer sharing weights with the first step. The proximal operators are non-linear activation layers and the gradient descent step is a weighted sum. We also rewrite the scalar multiplication as a $1 \times 1$ convolution. We refer the reader to Fig. 1 for an illustration. The lower figure depicts one iteration of inference while the whole inference process as a recurrent net is shown in the top figure.

Note that the whole inference process has two stages: first we compute the unaries $h(\mathbf{x}; \mathbf{w}_u)$ with a forward pass. Then we perform MAP inference through our recurrent network.

The first non-linearity for the primal dual method is the proximal operator of the dual function of $f_\alpha$. This changes for other types of proximal methods. In the case of the alternating direction method of multipliers (ADMM) the nonlinearity corresponds to the proximal operator of $f_\alpha$; for half-qudratic splitting it is the proximal operator of $f_\alpha$'s primal form while the second non linearity is a least-squares solver; if $f_i$ or $f_\alpha$ is reduced to a quadratic function of $\mathbf{y}$, the algorithm is simplified, as the proximal operator of a quadratic function is a linear function [5]. We refer the reader to the supplementary material for more details on other proximal methods.

## 2.4 Learning

Given training pairs composed of inputs $\{\mathbf{x}_n\}_{n=1}^N$ and their corresponding output $\{\mathbf{y}_n^{\text{gt}}\}_{n=1}^N$, learning aims at finding parameters which minimizes a regularized loss function:

$$
\mathbf{w}^* = \arg\min_{\mathbf{w}} \sum_n \ell(\mathbf{y}_n^*, \mathbf{y}_n^{\text{gt}}) + \gamma r(\mathbf{w})
$$

where $\ell(\cdot)$ is the loss, $r(\cdot)$ is a regularizer of the weight (we use $\ell_2$-norm in practice), $\mathbf{y}_n^*$ is the minimizer of Eq. (3) for the $n$-th example and $\gamma$ is a scalar. Given the conditions that both $\operatorname{prox}_{f_i}$ and $\operatorname{prox}_{g_\alpha}$ (or $\operatorname{prox}_{g_\alpha^*}$) are sub-differentiable wrt. $\mathbf{w}$ and $\mathbf{y}$, back-propagation can be used to compute the gradient efficiently. We refer the reader to Fig. 2 for an illustration of our learning algorithm.

Parameters such as the gradient steps $\sigma_\rho, \sigma_\tau, \sigma_{ex}$ in Eq. (5) are considered hyper-parameters in proximal methods and are typically manually set. In contrast, we can learn them as they are $1 \times 1$ convolution weights.

```
┌─────────────────────────────────────────────────────────────────────┐
│ Algorithm: Learning Continuous-Valued Deep Structured Models          │
│ Repeat until stopping criteria                                        │
│     1. Forward pass to compute $h_i(\mathbf{x}, \mathbf{w})$ and $h_\alpha(\mathbf{x}, \mathbf{w})$ │
│     2. Compute $\mathbf{y}^*$ i via forward pass in Eq. (5)            │
│     3. Compute the gradient via backward pass                         │
│     4. Parameter update                                               │
└─────────────────────────────────────────────────────────────────────┘
```

Figure 2: Algorithm for learning proximal deep structured models.

**Non-shared weights:** The weights and gradient steps for high-order potentials are shared among all the iteration blocks in the inference network, which guarantees the feed-forward pass to explicitly minimize the energy function in Eq. (2). In practice we found that by removing the weight-sharing and fixed gradient step constraints, we can give extra flexibility to our model, boosting the final performance. This observation is consistent with the findings of shrinkage field [30] and inference machines [27].

**Multi-loss:** Intermediate layer outputs $\mathbf{y}^{(t)}$ should gradually converge towards the final output. Motivated by this fact, we include a loss over the intermediate computations to accelerate convergence.

### 2.5 Discussion and Related Work

Our approach can be considered as a continuous-valued extension of deep structured models [3, 31, 39]. Unlike previous methods where the output lies in a discrete domain and inference is conducted through a specially designed message passing layer, the output of the proposed method is in continuous domain and inference is done by stacking convolution and non-linear activation layers. Without deep unary potentials, our model is reduced to a generalized version of field-of-experts [28]. The idea of stacking shrinkage functions and convolutions as well as learning iteration-specific weights was exploited in the learning iterative shrinkage algorithm (LISTA) [14]. LISTA can be considered as a special case of our proposed model with sparse coding as the energy function and proximal gradient as the inference algorithm. Our approach is also closely related to the recent structured prediction energy networks (SPEN) [1], where our unary network is analogous to the feature net in SPEN and the whole energy model is analogous to the energy net. Both SPEN and our proposed method can be considered as a special case of optimization-based learning [8]. However, SPEN utilizes plain gradient descent for inference while our network is proximal algorithm motivated. Previous methods have tried to learn multi-variate regression networks for optical flow [11] and stereo [24]. But none of these approaches model the interactions between output variables. Thus, they can be considered a special case of our model, where only unary functions $f_i$ are present.

## 3 Experiments

We demonstrate the effectiveness of our approach in three different applications: image denoising, depth refinement and optical flow estimation. We employ mxnet [4] with CUDNNv4 acceleration to implement the networks, which we train end-to-end. Our experiments are conducted on a Xeon 3.2 Ghz machine with a Titan X GPU.

### 3.1 Image Denoising

We first evaluate our method for the task of image denoising (*i.e.*, shallow unary) using the BSDS image dataset [23]. We corrupt each image with Gaussian noise with standard deviation $\sigma = 25$. We use the energy function typically employed for image denoising:

$$\mathbf{y}^* = \arg\min_{\mathbf{y} \in \mathcal{Y}} \sum_i \|y_i - x_i\|_2^2 + \lambda \sum_\alpha \|\mathbf{w}_{\text{ho},\alpha}^T \mathbf{y}_\alpha\|_1 \tag{6}$$

According to the primal dual algorithm, the activation function for the first nonlinearity is the proximal operator of the dual function of the $\ell_1$-norm: $\text{prox}_\rho^*(z) = \min(|z|, 1) \cdot \text{sign}(z)$, which is the projection onto an $\ell_\infty$-norm ball. In practice we encode this function as $\text{prox}_\rho^*(z) = \max(\min(z, 1), -1)$. The

|               | BM3D [6] | EPLL [40] | LSSC [22] | CSF [30] | RTF [29] | Ours     | Ours GPU |
|---------------|----------|-----------|-----------|----------|----------|----------|----------|
| PSNR          | 28.56    | 28.68     | 28.70     | 28.72    | 28.75    | **28.79**| **28.79**|
| Time (second) | 2.57     | 108.72    | 516.48    | 5.10     | 69.25    | **0.23** | **0.011**|

Table 1: Natural Image Denoising on BSDS dataset [23] with noise variance $\sigma = 25$.

|    | $3 \times 3$ | $5 \times 5$ | $7 \times 7$ |
|----|--------------|--------------|--------------|
| 16 | 28.43        | 28.57        | 28.68        |
| 32 | 28.48        | 28.64        | 28.76        |
| 64 | 28.49        | 28.68        | 28.79        |

Table 2: Performance of the proposed model with different hyper-parameters

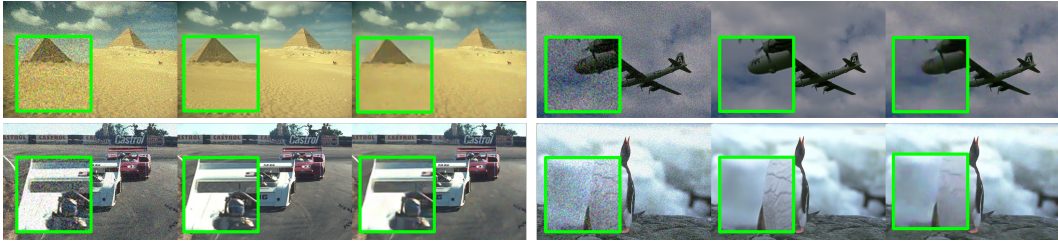

Figure 3: Qualitative results for image denoising. Left to right: noisy input, ground-truth, our result.

second nonlinearity is the proximal operator of the primal function of the $\ell_2$-norm, which is a weighted sum: $\mathrm{prox}_{\ell_2}(y, \lambda) = \frac{x + \lambda y}{1 + \lambda}$.

For training, we select 244 images, following the configuration of [30]. We randomly cropped $128 \times 128$ clean patches from the training images and obtained the noisy input by adding random noise. We use mean square error as the loss function and set a weight decay strength of $0.0004$ for all settings. Note that for all the convolution and deconvolution layers, the bias is set to zero. MSRA initialization [16] is used for the convolution parameters and the initial gradient step for each iteration is set to be $0.02$. We use adam [19] with a learning rate of $t = 0.02$ and hyper-parameters $\beta_1 = 0.9$ and $\beta_2 = 0.999$ as in Kingma *et al*. [19]. The learning rate is divided by 2 every 50 epoch, and we use a mini-batch size of 32.

We compare against a number of recent state-of-the-art techniques [6, 40, 22, 30, 29]. [2] The Peak Signal-to-Noise Ratio (PSNR) is used as a performance measure. As shown in Tab. 1 our proposed method outperforms all methods in terms of accuracy and speed. The second best performing method is RTF [29], while being two orders of magnitude slower than our approach. Our GPU implementation achieves real-time performance with more than 90 frames/second. Note that a GPU version of CSF is reported to run at $0.92s$ on a $512 \times 512$ image on a GTX 480 [30]. However, since GPU implementation is not available online, we cannot make proper comparisons.

Tab. 2 shows performance with different hyper-parameters (filter size, number of filters per each layer). As we can see, larger receptive fields and more convolution filters slightly boost the performance. Fig. 3 depicts the qualitative results of our model for the denoising task.

### 3.2 Depth Refinement

Due to specularities and intensity changes of structured light imaging, the sensor's output depth is often noisy. Thus, refining the depth to generate a cleaner, more accurate depth image is an important task. We conduct the depth refinement experiment on the 7 Scenes dataset [25]. We follow the configuration of [10], where the ground-truth depth was computed using KinectFusion [25]. The noise [10] has a Poisson-like distribution and is depth-dependent, which is very different from the image denoising experiment which contained Gaussian noise.

We use the same architecture as for the task of natural image denoising. The multi-stage mean square error is used as loss function and the weight decay strength is set to be $0.0004$. Adam ($\beta_1 = 0.9$ and

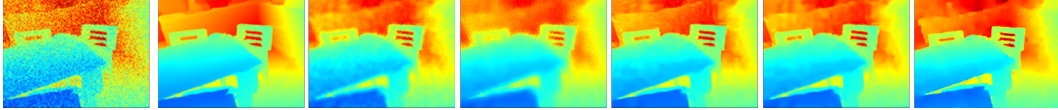

Figure 4: Qualitative results for depth refinement. Left to right: input, ground-truth, wiener filter, bilateral filter, BM3D, Filter Forest, Ours.

|       | Wiener | Bilateral | LMS   | BM3D [6] | FilterForest [10] | Ours      |
|-------|--------|-----------|-------|----------|-------------------|-----------|
| PSNR  | 32.29  | 30.95     | 24.37 | 35.46    | 35.63             | **36.31** |

Table 3: Performance of depth refinement on dataset [10]

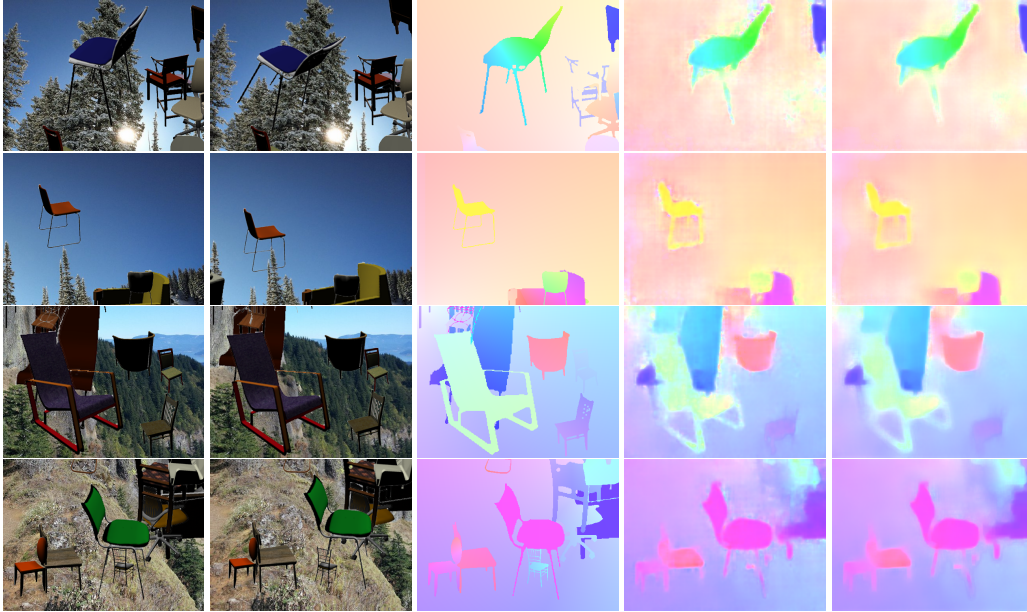

Figure 5: **Optical flow:** Left to right: first and second input, ground-truth, Flownet [11], ours.

$\beta_2 = 0.999$) is used as the optimizer with a learning rate of $0.01$. Data augmentation is used to avoid overfitting, including random cropping, flipping and rotation. We used a mini-batch size of 16.

We train our model on 1000 frames of the Chess scene and test on the other scenes. PSNR is used to evaluate the performance. As shown in Tab. 3, our approach outperforms all competing algorithms. This shows that our deep structured network is able to handle non-additive non-Gaussian noise. Qualitative results are shown in Fig. 4. Compared to the competing approaches, our method is able to recover better depth estimates particularly along the depth discontinuities.

### 3.3 Optical Flow

We evaluate the task of optical flow estimation on the Flying Chairs dataset [11]. The size of training images is $512 \times 384$. We formulate the energy as follows:

$$\mathbf{y}^* = \arg\min_{\mathbf{y} \in \mathcal{Y}} \sum_i \|y_i - f_i(\mathbf{x}^l, \mathbf{x}^r; \mathbf{w}_u)\|_1 + \lambda \sum_\alpha \|\mathbf{w}_{\text{ho},\alpha}^T \mathbf{y}_\alpha\|_1 \qquad (7)$$

where $f_i(\mathbf{x}^l, \mathbf{x}^r; \mathbf{w}_u)$ is a Flownet model [11], is a fully-convolutional encoder-decoder network that predicts 2D optical flow per pixel. It has 11 encoding layers and 11 deconv layers with skip connections. $\mathbf{x}^l$ and $\mathbf{x}^r$ are the left and right input images respectively and $\mathbf{y}$ is the desired optical flow output. Note that we use the $\ell_1$-norm for both, the data and the regularization term. The first nonlinearity activation function is the proximal operator of the $\ell_1$-norm's dual function: $\text{prox}_\rho^*(z) = \min(|z|, 1) \cdot \text{sign}(z)$, and the second non-linear activation function is the proximal operator of the $\ell_1$-norm's primal form: $\text{prox}_{\tau,x}(y, \lambda) = x - \min(|x - y|, \lambda) \cdot \text{sign}(x)$, which is a soft shrinkage function [26].

| | Flownet | Flownet + TV-l1 | Our proposed |
|---|---|---|---|
| End-point-error | 4.98 | 4.96 | **4.91** |

Table 4: Performance of optical flow on Flying chairs dataset [11]

We build a deep structured model with 5 iteration blocks. Each iteration block has 32 convolution filters of size $7 \times 7$ for both the convolution and deconvolution layers, which results in 10 convolution/deconv layers and 10 non-linearities. The multi-stage mean square error is used as the loss function and the weight decay strength is set to be 0.0004.

Training is conducted on the training subset of the Flying Chairs dataset. Our unary model is initialized with a pre-trained Flownet parameters. The high-order term is initialized with MSRA random initialization [16]. The hyper-parameter $\lambda$ in this experiment is pre-set to be 10. We use random flipping, cropping and color-tuning for data augmentation, and employ the adam optimizer with the same configuration as before ($\beta_1 = 0.9$ and $\beta_2 = 0.999$) with a learning rate $t = 0.005$. The learning rate is divided by 2 every 10 epoch and the mini-batch size is set to be 12.

We evaluate all approaches on the test set of the Flying chairs dataset. End-point error is used as a measure of performance. The unary-only model (*i.e.* plain flownet) is used as baseline and we also compare against a plain TV-l1 model with four pre-set gradient operators as post-processing. As shown in Tab. 4 our method outperforms all the baselines. From Fig. 5 we can see that our method is less noisy than Flownet's output and better preserves the boundaries. Note that our current model is isotropic. In order to further boost the performance, incorporating anisotropic filtering like bilateral filtering is an interesting future direction.

## 4 Conclusion

We have proposed a deep structured model that learns non-linear functions encoding complex dependencies between continuous output variables. We have showed that inference in our model using proximal methods can be efficiently solved as a feed-foward pass on a special type of deep recurrent neural network. We demonstrated our approach in the tasks of image denoising, depth refinement and optical flow. In the future we plan to investigate other proximal methods and a wider variety of applications.

## Footnotes

[1] A function $g : \mathcal{Y} \times \mathcal{Y} \to [0, \infty)$ is called a distance function iff it satisfies the condition of non-negativity, identity of indisernibles, symmetry and triangle inequality.

[2] We chose the model with the best performance for each competing algorithm. For the CSF method, we use $\mathrm{CSF}^5_{7 \times 7}$; for RTF we use $\mathrm{RTF}_5$; for our method, we pick $7 \times 7 \times 64$ high-order structured network.

## References

[1] David Belanger and Andrew McCallum. Structured prediction energy networks. In *ICML*, 2016.

[2] A. Chambolle and T. Pock. A first-order primal-dual algorithm for convex problems with applications to imaging. *JMIV*, 2011.

[3] L. Chen, A. Schwing, A. Yuille, and R. Urtasun. Learning deep structured models. In *ICML*, 2015.

[4] T. Chen, M. Li, Y. Li, M. Lin, N. Wang, M. Wang, T. Xiao, B. Xu, C. Zhang, and Z. Zhang. Mxnet: A flexible and efficient machine learning library for heterogeneous distributed systems. *arXiv*, 2015.

[5] Y. Chen, W. Yu, and T. Pock. On learning optimized reaction diffusion processes for effective image restoration. In *CVPR*, 2015.

[6] K. Dabov, A. Foi, V. Katkovnik, and K. Egiazarian. Image denoising by sparse 3-d transform-domain collaborative filtering. *TIP*, 2007.

[7] J. Deng, N. Ding, Y. Jia, A. Frome, K. Murphy, S. Bengio, Y. Li, H. Neven, and H. Adam. Large-scale object classification using label relation graphs. In *ECCV*. 2014.

[8] Justin Domke. Generic methods for optimization-based modeling. In *AISTATS*, 2012.

[9] D. Eigen and R. Fergus. Predicting depth, surface normals and semantic labels with a common multi-scale convolutional architecture. In *ICCV*, 2015.

[10] S. Fanello, C. Keskin, P. Kohli, S. Izadi, J. Shotton, A. Criminisi, U. Pattacini, and T. Paek. Filter forests for learning data-dependent convolutional kernels. In *CVPR*, 2014.

[11] P. Fischer, A. Dosovitskiy, E. Ilg, P. Häusser, C. Hazırbaş, V. Golkov, P. van der Smagt, D. Cremers, and T. Brox. Flownet: Learning optical flow with convolutional networks. In *CVPR*, 2015.

[12] D. Gabay and B. Mercier. A dual algorithm for the solution of nonlinear variational problems via finite element approximation. *Computers & Mathematics with Applications*, 1976.

[13] D. Geman and C. Yang. Nonlinear image recovery with half-quadratic regularization. *TIP*, 1995.

[14] Karol Gregor and Yann LeCun. Learning fast approximations of sparse coding. In *ICML*, 2010.

[15] K. He, X. Zhang, S. Ren, and J. Sun. Deep residual learning for image recognition. *arXiv*, 2015.

[16] K. He, X. Zhang, S. Ren, and J. Sun. Delving deep into rectifiers: Surpassing human-level performance on imagenet classification. In *ICCV*, 2015.

[17] G. Hinton, L. Deng, D. Yu, G. Dahl, A. Mohamed, N. Jaitly, A. Senior, V. Vanhoucke, P. Nguyen, T. Sainath, et al. Deep neural networks for acoustic modeling in speech recognition. *SPM, IEEE*, 2012.

[18] A. Ihler and D. McAllester. Particle belief propagation. In *AISTATS*, 2009.

[19] D. Kingma and J. Ba. Adam: A method for stochastic optimization. *arXiv*, 2014.

[20] D. Krishnan and R. Fergus. Fast image deconvolution using hyper-laplacian priors. In *NIPS*, 2009.

[21] A. Krizhevsky, I. Sutskever, and G. Hinton. Imagenet classification with deep convolutional neural networks. In *NIPS*, 2012.

[22] J. Mairal, F. Bach, J. Ponce, G. Sapiro, and A. Zisserman. Non-local sparse models for image restoration. In *ICCV*, 2009.

[23] D. Martin, C. Fowlkes, D. Tal, and J. Malik. A database of human segmented natural images and its application to evaluating segmentation algorithms and measuring ecological statistics. In *ICCV*, 2001.

[24] N. Mayer, E. Ilg, P. Häusser, P. Fischer, D. Cremers, A. Dosovitskiy, and T. Brox. A large dataset to train convolutional networks for disparity, optical flow, and scene flow estimation. *arXiv*, 2015.

[25] R. Newcombe, S. Izadi, O. Hilliges, D. Molyneaux, D. Kim, A. Davison, P. Kohi, J. Shotton, S. Hodges, and A. Fitzgibbon. Kinectfusion: Real-time dense surface mapping and tracking. In *ISMAR*, 2011.

[26] N. Parikh and S. Boyd. Proximal algorithms. *Foundations and Trends in optimization*, 2014.

[27] S. Ross, D. Munoz, M. Hebert, and J. Bagnell. Learning message-passing inference machines for structured prediction. In *CVPR*, 2011.

[28] S. Roth and M. Black. Fields of experts: A framework for learning image priors. In *CVPR*, 2005.

[29] U. Schmidt, J. Jancsary, S. Nowozin, S. Roth, and C. Rother. Cascades of regression tree fields for image restoration. *PAMI*, 2013.

[30] U. Schmidt and S. Roth. Shrinkage fields for effective image restoration. In *CVPR*, 2014.

[31] A. Schwing and R. Urtasun. Fully connected deep structured networks. *arXiv*, 2015.

[32] E. Sudderth, A. Ihler, M. Isard, W. Freeman, and A. Willsky. Nonparametric belief propagation. *Communications of the ACM*, 2010.

[33] I. Sutskever, O. Vinyals, and Q. Le. Sequence to sequence learning with neural networks. In *NIPS*, 2014.

[34] I. Tsochantaridis, T. Hofmann, T. Joachims, and Y. Altun. Support vector machine learning for interdependent and structured output spaces. In *ICML*, 2004.

[35] S. Wang, A. Schwing, and R. Urtasun. Efficient inference of continuous markov random fields with polynomial potentials. In *NIPS*, 2014.

[36] Y. Weiss and W. Freeman. Correctness of belief propagation in gaussian graphical models of arbitrary topology. *Neural computation*, 2001.

[37] C. Zach and P. Kohli. A convex discrete-continuous approach for markov random fields. In *ECCV*. 2012.

[38] J. Zbontar and Y. LeCun. Computing the stereo matching cost with a convolutional neural network. In *CVPR*, 2015.

[39] S. Zheng, S. Jayasumana, B. Romera-Paredes, V. Vineet, Z. Su, D. Du, C. Huang, and P. Torr. Conditional random fields as recurrent neural networks. In *ICCV*, 2015.

[40] D. Zoran and Y. Weiss. From learning models of natural image patches to whole image restoration. In *ICCV*, 2011.

